# Dual-Diffusion for Binocular 3D Human Pose Estimation

**Xiaoyue Wan**    **Zhuo Chen**    **Bingzhi Duan**    **Xu Zhao** *
Department of Automation
Shanghai Jiao Tong University
{sherrywaan, chzh9311, DuanBingzhi, zhaoxu}@sjtu.edu.cn

## Abstract

Binocular 3D human pose estimation (HPE), reconstructing a 3D pose from 2D poses of two views, offers practical advantages by combining multiview geometry with the convenience of a monocular setup. However, compared to a multiview setup, the reduction in the number of cameras increases uncertainty in 3D reconstruction. To address this issue, we leverage the diffusion model, which has shown success in monocular 3D HPE by recovering 3D poses from noisy data with high uncertainty. Yet, the uncertainty distribution of initial 3D poses remains unknown. Considering that 3D errors stem from 2D errors within geometric constraints, we recognize that the uncertainties of 3D and 2D are integrated in a binocular configuration, with the initial 2D uncertainty being well-defined. Based on this insight, we propose **Dual-Diffusion** specifically for Binocular 3D HPE, simultaneously denoising the uncertainties in 2D and 3D, and recovering plausible and accurate results. Additionally, we introduce Z-embedding as an additional condition for denoising and implement baseline-width-related pose normalization to enhance the model flexibility for various baseline settings. This is crucial as 3D error influence factors encompass depth and baseline width. Extensive experiments validate the effectiveness of our Dual-Diffusion in 2D refinement and 3D estimation. The code and models are available at https://github.com/sherrywan/Dual-Diffusion.

## 1 Introduciton

3D human pose estimation (HPE) aims to localize the 3D position of human joints, which has a broad range of downstream applications [49]. To date, monocular 3D HPE [25, 30, 20, 60, 22, 35, 59, 58] has received a great deal of attention due to its convenient for practical applications, while multiview (more than two cameras) 3D HPE [31, 14, 56, 5, 46] has earned popularity due to its absolute localization under geometric constraints. However, the pros and cons of these two setups are "conjugat" to each other, with monocular suffering from depth ambiguity, while multiview is hindered by strict scene constraints. Binocular setup [45] offers both advantages, yet has long been ignored by the community. This motivates us to focus on the Binocular 3D HPE, which lifts to a 3D pose from binocular 2D poses.

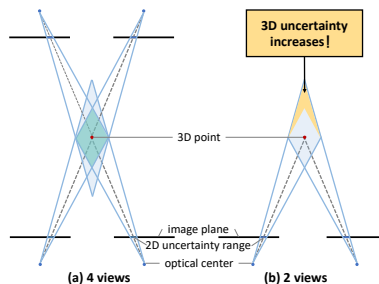

Figure 1: Binocular reconstruction has higher 3D uncertainty compared to multiview configurations.

Observing the uncertainty range of a 3D point reconstructed under different numbers of cameras, shown in Fig. 1, it is evident that although binocular setups significantly reduce 3D reconstruction uncertainty compared to monocular configurations due to geometric constraints, they still have higher ambiguity compared to multiview setups. In previous works on monocular 3D HPE, to alleviate ambiguity, human pose priors such as joint angle limits [1] and physical plausibility [54] are commonly modeled. Nowadays, with advancements in probabilistic methods in machine learning, many works aim to leverage data to model the distribution of real human poses as a representation of pose prior, including VAE [23, 29, 48], GAN [47, 7], normalizing flow [19, 50], and diffusion models [8, 36, 16]. Among these, diffusion models stand out due to their advantages like indirect likelihood correlation, simple training, and network flexibility [12, 38]. *Given the success of diffusion models in modeling pose priors, can they be cleverly leveraged to reduce uncertainty in binocular 3D HPE?*

The diffusion model [11] comprises two processes: the forward diffusion process, which perturbs the real data to a diffused distribution; and the reverse denoising process, which denoises the noisy data to match the real distribution. In monocular 3D HPE, the diffusion model is employed to recover 3D poses from noisy data sampled from random noise [34, 12] or from an initial 3D pose distribution with high uncertainty [8]. The first category is excluded due to high time consumption, as it fails to leverage geometric constraints [9] to narrow down the search space. The second category starts from the initial 3D poses with higher uncertainty (estimated by off-the-shelf 3D HPE methods) and converges to the lower uncertainty distribution representative of more plausible and accurate 3D poses, which is more efficient. However, the main problem lies in the unknown distribution of the initial 3D uncertainty. The statistical method is used in [8] to solve it which is unrealistic in practice. Unlike monocular setups, 3D errors stem from errors in 2D estimation in binocular within a geometric framework. In other words, 3D uncertainty can be reconstructed from 2D cause they are intrinsically linked by geometric constraints. *Therefore, we propose a diffusion-based method specifically for binocular 3D HPE, named Dual-Diffusion, capable of simultaneously denoising initial 2D and 3D uncertainties.*

From a diffusion perspective, the uncertainty distribution in initial 2D joints is well-defined, typically modeled as a Gaussian distribution centered at the ground truth with a specified standard deviation [26, 51, 40]. However, the uncertainty in initial 3D poses remains unknown. Nevertheless, from a denoising standpoint, it is preferable to recover poses in 3D space [6, 61] rather than in 2D, given that human pose priors are inherently 3D. To bridge the gap between diffusion and denoising across different domains, we leverage geometric projection techniques to couple the 2D plane and 3D space. The specific modeling of Dual-Diffusion is illustrated in Fig. 2(a). In the forward diffusion process, noisy 2D binocular poses are generated from ground truth 2D poses by adding noise step-by-step. Subsequently, Triangulation [9] is used to reconstruct noisy 3D poses, thereby defining the uncertainty distribution in 3D. During the reverse denoising process, we sample noisy 3D poses and remove the noise to recover plausible and accurate 3D poses. Reprojection is utilized to estimate the corresponding 2D poses back, enabling simultaneous denoising in both 2D and 3D spaces.

Reviewing our Dual-Diffusion model, the essential objective of the denoiser network is to remove the noise of noisy 3D poses under different perturbed distributions. The noise-perturbed level is determined by the noise addition step $t$ in diffusion and is reflected in the denoiser via timestamp embedding. While $t$ is directly associated with 2D noise, 3D noise depends not only on 2D factors but also on 3D depth and the baseline width of the binocular setup. To enable flexible denoising of 3D noisy data under the same $t$ but with varying uncertainties, we introduce 3D depth as an additional condition, named Z-embedding. Besides, the baseline-width-related normalization (BaseL-norm) is applied to 3D poses, which allows the model to flexibly adapt to different baseline width settings.

To validate the efficacy of our Dual-Diffusion model in denoising both 2D and 3D poses, we conducted experiments on the binocular H36M [13] and MHAD [27] datasets, utilizing only 2-view camera pairs. Our model outperforms the baseline Triangulation and other state-of-the-art methods. Additionally, we establish the "random-noise" and "2D-Diff" models for comparison, demonstrating the effectiveness of starting from initial pose distributions and leveraging 3D pose priors.

Our contributions can be summarized as follows:

- **Dual-Diffusion Framework**. We propose a novel framework, Dual-Diffusion, specifically designed for binocular 3D HPE. This model simultaneously removes noise from both 2D and 3D poses by leveraging geometric mapping.

- **Uncertainty Analysis and Denoiser Enhancement**. We analyze the relationship between 3D and 2D uncertainties and introduce Z-embedding and BaseL-norm to enhance the flexibility of the denoiser.
- **Benchmark Performance**. Our method achieves superior performance on the evaluated benchmarks, demonstrating effectiveness.

## 2   Related Work

**Binocular and Multiview 3D HPE.**   Binocular 3D HPE aims to estimate 3D poses from single-frame 2D poses captured from two perspectives. This task has been sparsely studied. RSB-Pose [45] utilizes stereo volume features to enhance binocular coherence in 2D poses and employs a spatial Transformer for 3D pose refinement. While the Transformer excels at establishing correlations between nodes, its ability to effectively identify and denoise incorrect joints is uncertain. Given the geometric constraints shared in multiview setups, we review multiview 3D HPE methods, typically involving two stages: 1) estimating 2D poses from images, and 2) lifting these to 3D poses using geometric constraints, with Triangulation [9] being the most common approach. Existing methods can be categorized into two stages of improvement. The first category [32, 10, 33, 46] enhances 2D poses using 3D-aware features from multiview fusion. The second category [3, 31, 14, 5, 15] focuses on lifting 2D poses to 3D. Explicit pose priors, such as bone length, are incorporated into the lifting process using Pictorial Structure Models [3, 31] or closed-form Structural Triangulation [5]. Some methods [14, 15] treat lifting as a 3D regression task using volume representations, implicitly considering pose coherence. However, these approaches either rely on limited explicit pose priors or involve computationally intensive 3D convolutions. Actually, due to geometric constraints, the 3D poses reconstructed from initial 2D poses in multiview are generally reliable, with limited studies focusing on 2D-3D lifting. Recognizing that the primary challenge in binocular setups is increased 3D uncertainty, this work specifically focuses on the 2D-3D lifting process. Hence, we review monocular 3D HPE methods for additional insights.

**Monocular 3D HPE.**   Monocular 3D HPE methods can be categorized into one-stage and two-stage approaches. One-stage methods [30, 43, 62, 42] directly regress the 3D pose from the image, relying on extensive datasets and complex network architectures. Two-stage [25, 18, 4] first estimate the 2D pose using a 2D detector, then lift it to 3D through deep network mapping. Considering the joints are connected within a skeleton, Graph Convolutional Networks (GCNs) [57, 2, 60] and Transformer architectures [35, 55, 53, 41, 59] are introduced to establish correlations between joints during the lifting process. However, despite establishing joint relationships, monocular methods still suffer from inherent depth ambiguity, which utilizes temporal consistency or poses priors to overcome.

**Pose Priors in 3D HPE.**   Since our task involves single-frame pose estimation, we primarily review works focusing on modeling pose priors in monocular 3D HPE. Previous approaches explore explicit pose priors, such as joint angle limits [1], physical plausibility [54], or defined pose models [21]. However, explicit priors, despite their interpretability, may not be comprehensive. With advancements in image generation, probabilistic methods capable of modeling data distributions gain attention. VAE [23, 29, 48] encodes poses into a latent space following a normal distribution, then decodes them back to the original pose. However, inference through recurrently optimizing latent parameters can be time-consuming. Normalizing flows [19, 50] use invertible transformations to map latent features to 3D poses, aiding inference, but with complex network architectures. Generative adversarial networks (GANs) [47, 7] learn pose distributions by distinguishing fake and real poses, but face challenges in training. The diffusion models [11, 38, 39] gain popularity due to its network flexibility and training simplicity. It learns data distribution by iteratively removing noise which is incrementally added to real data until a fully diffused noise distribution is achieved. Several methods [6, 12, 36, 34, 61] condition the 3D pose distribution on estimated 2D points. However, directly applying this framework to binocular 3D HPE disregards geometric constraints, acquiring more diffusion and denoising steps. Another approach [8] treats initial 3D poses as containing high uncertainty and accurate 3D poses as maintaining low uncertainty, and employs the diffusion model to denoise noisy 3d pose from an initial high-uncertainty distribution to a more accurate result under a certain distribution. However, the initial distribution of 3D poses is unknown which is estimated statistically in [8]. Considering geometric constraints in binocular configuration, the initial 3D pose distribution can be derived from

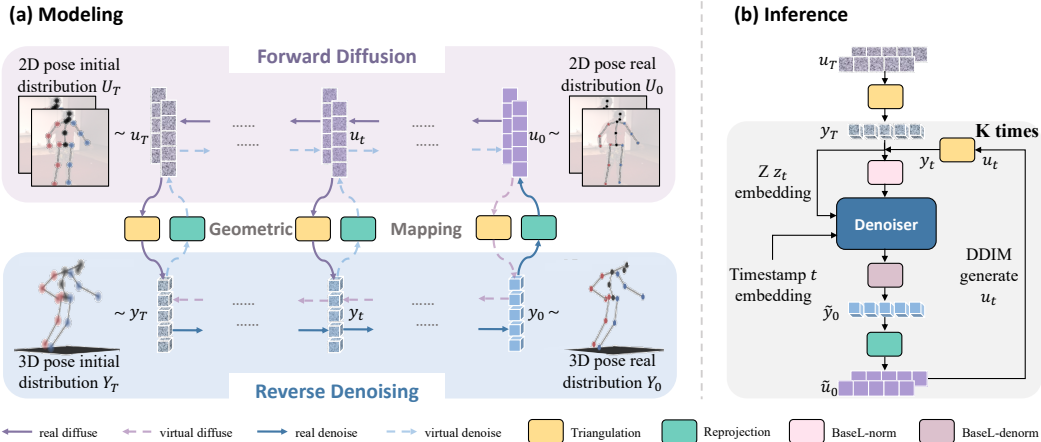

Figure 2: **Overview of Dual-Diffusion Method.** (a) Modeling: In the forward diffusion process, noise is added to the ground truth binocular 2D poses $u_0$ for $T$ steps, aligning with the distribution of initial estimated 2D poses. During the reverse denoising process, noisy 3D poses are progressively denoised to plausible poses. Geometric mapping is employed to connect 2D and 3D domains. (b) Inference: The initial 3D pose $y_T$, reconstructed from binocular 2D poses $u_T$, is denoised to $\tilde{y}_0$. Then $\tilde{y}_0$ is reprojected to the denoised 2D poses $\tilde{u}_0$. The entire denoising process iterates for $K$ times.

the initial 2D distribution which is more straightforward. Therefore, we propose the Dual-Diffusion model to denoise both 2D and 3D poses, specifically for binocular 3D HPE.

## 3 Method

We propose Dual-Diffusion to simultaneously optimize 2D poses and 3D poses, addressing the high uncertainty issue in binocular 3D HPE. The modeling framework is illustrated in Fig. 2(a). Starting with the initial 2D poses estimated by the off-the-shelf 2D detector, which are considered as the diffused data, we apply a diffusion model to denoise it. However, the denoising is applied to the 3D pose rather than the 2D pose, given the inherently 3D nature of pose priors. The geometric projection relationship is leveraged to bridge the gap and enable dual-denoising of 2D and 3D space. In the subsequent sections, we first briefly introduce the diffusion models and then describe our Dual-Diffusion model in detail.

### 3.1 Revisiting Diffusion Models

Diffusion models [11, 38] are a type of probabilistic method that can recover data that satisfy the underlying distribution $p_{data}(x)$ from noisy data. It comprises two processes: the forward diffusion process and the reverse denoising process. During the forward process, the real data $x_0$ is diffused by a Gaussian noise step-by-step over $T$ steps under a Markov chain. The formulation can be written as:

$$q(x_{1:T}|x_0) := \prod_{t=1}^{T} q(x_t|x_{t-1}), \quad q(x_t|x_{t-1}) := \mathcal{N}(x_t; \sqrt{1-\beta_t}x_{t-1}, \beta_t\mathbf{I}), \tag{1}$$

where the variance schedule $\beta_t$ determines the perturbed level of the noisy data $x_t$. Using the notation $\alpha_t := 1 - \beta_t$ and $\bar{\alpha}_t := \prod_{t=1}^{T} \alpha_t$, $x_t$ can be sampled from $x_0$ skipping timestamps $1 : t - 1$:

$$q(x_t|x_0) = \mathcal{N}(x_t; \sqrt{\bar{\alpha}_t}x_0, (1-\bar{\alpha}_t)\mathbf{I}) \quad \text{and} \quad x_t = \sqrt{\bar{\alpha}_t}x_0 + \sqrt{1-\bar{\alpha}_t}\epsilon_t, \epsilon_t \sim \mathcal{N}(0, \mathbf{I}). \tag{2}$$

When $T$ is large enough, $x_T \sim \mathcal{N}(0, \mathbf{I})$ can be satisfied. Hence, the reverse process starts at samples from $\mathcal{N}(0, \mathbf{I})$, and the purpose is to recover $x_0 \sim p_{data}(x)$. According to ELBO [17], the target $\max \log p_\theta(x_0)$ can be simplified to minimize the KL divergence between the reverse conditional distribution $p_\theta(x_{t-1}|x_t)$ and the posterior of the diffusion $q(x_{t-1}|x_t, x_0)$ which is formulated as:

$$q(x_{t-1}|x_t, x_0) = \mathcal{N}(x_{t-1}; \tilde{\mu}_t(x_t, x_0), \tilde{\beta}_t\mathbf{I}), \quad \tilde{\beta}_t := \frac{1-\bar{\alpha}_{t-1}}{1-\bar{\alpha}_t}, \tag{3}$$

where $\tilde{\mu}_t$ is a linear combination of $x_t$ and $x_0$. The reverse conditional distribution can be ensured in Gaussian form $p_\theta(x_{t-1}|x_t) \sim \mathcal{N}(x_{t-1}; \mu_\theta(x_t, t), \Sigma_\theta(x_t, t))$ if $\beta_t$ are small. Thus, the key to KL divergence is the $L_2$ distance between $\mu_\theta$ and $\tilde{\mu}_t$, which drives the denoiser network to learn $\mu_\theta$ to predict $\tilde{\mu}_t$. According to Eq. 2, the loss of the denoiser training is finally simplified to:

$$L_\theta = \mathbb{E}_{t,x_0,\epsilon_t}[\|\epsilon_t - \epsilon_\theta(\sqrt{\bar{\alpha}_t}x_0 + \sqrt{1 - \bar{\alpha}_t}\epsilon_t, t)\|^2]. \tag{4}$$

The essential of the denoiser is to predict the noise $\epsilon_t$ at any perturbed level $t$ added to the real data, and then can recover the $x_0$.

## 3.2 Dual Diffusion

Starting from $\mathcal{N}(0, \mathbf{I})$ is highly time-consuming due to the high diffusion of noise. [8] suggests that the 3D pose $y_T$ predicted by the off-the-shelf methods is a kind of noisy data under the distribution with high uncertainty $\mathcal{Y}_T$, and diffusion models can be used to reduce uncertainty to generate the accurate result $y_0$. Starting from $\mathcal{Y}_T$ with limited diffusion is more efficient. The major problem is that the 3D pose initial uncertainty is unknown, which is solved by the statistical method in [8]. However, if the uncertainty of the 3D pose follows a Gaussian distribution is still confusing. And the statistical results can be easily influenced by the models and training dataset, posing challenges in practical applications. In this work, considering the binocular geometry framework, we design an elegant method to build uncertainty distribution of initial 3D poses from 2D uncertainty, and conversely, to achieve 2D denoising by denoising 3D poses, which we name Dual-Diffusion.

**Forward Dual Diffusion** process gradually adds noise to ground truth binocular 2D poses $u_0 = \{u_{0,v}\} \in \mathbb{R}^{J \times 4}$ in a Markov chain. Here, $J$ is the number of joints and $v \in 0, 1$ indicates the left and the right perspectives. The diffusion domain is localized in 2D based on two reasonable assumptions: 1) the uncertainty distribution of initial 2D poses is known, and 2) it is in Gaussian form. We predict the initial 2D poses using a 2D detector. The training objective of this detector is to predict heatmaps of 2D joints whose supervision is a Gaussian distribution centered at the ground truth with a fixed standard deviation $\sigma_T$. The maturity of 2D detectors ensures the quality of the heatmap generation. Therefore, the initially predicted binocular 2D poses can be treated as the noisy data following the Gaussian distribution $\mathcal{N}(0, \sigma_T^2\mathbf{I})$ around the ground truth 2D poses $u_0$. This indicates that the diffusion target at $T$ step should satisfy the distribution $\mathcal{U}_T \sim \mathcal{N}(u_0, \sigma_T^2\mathbf{I})$. The formulation to create the diffused 2D poses can be written as:

$$u_t = u_0 + \sqrt{(1 - \bar{\alpha}_t)}\epsilon_t, \quad \epsilon_t \sim \mathcal{N}(0, \sigma_T^2\mathbf{I}). \tag{5}$$

Here, $x_0$ and $x_t$ in Eq. 2 are replaced by $u_0 - u_0$ and $u_t - u_0$ correspondingly to maintain the distribution consistency of $\epsilon_t$. When $t = T$, $u_T \sim \mathcal{U}_T$ is satisfied. The diffusion is transmitted to 3D space by Triangulation [9], which is a geometric tool to reconstruct 3D poses $y \in \mathbb{R}^{J \times 3}$ from binocular 2D poses using camera parameters. The diffused 3D poses can be generated by:

$$y_t = Tri(u_{t,0}, u_{t,1}), \tag{6}$$

where $Tri$ represents the closed-form solver using Linear Triangulation. Up to now, the uncertainty distribution of initial 2D poses is determined. Although the initial uncertainty distribution $\mathcal{Y}_T$ in 3D has not yet been formulated, it can be reconstructed by sampling from noisy 2D poses.

**Reverse Dual Denoising** should recover accurate binocular 2D poses $u_0$ from the estimated 2D poses $u_T$ according to the definition in diffusion models. However, we argue that 3D poses inherently exhibit a more constrained distribution compared to 2D poses. 2D poses may be various under different camera parameters and perspectives. Hence, we modify the denoising network to recover the original 3D poses $y_0$. As above, the denoising is transmitted to the 2D plane by Reprojection, given the camera extrinsic and intrinsic matrix. Then the noise $\epsilon_t$ added to 2D poses in Eq. 5 can be predicted. However, taking into account the depth ambiguity problem in the correspondence between 3D and 2D noises, the denoiser objective in Eq. 4 is adjusted to predict the original data rather than noise, as stated:

$$L_\theta = \mathbb{E}_{t,u_{0,0},u_{0,1}}[\sum_v \|u_{0,v} - Rpj_v(y_\theta)\|^2], \quad y_\theta = f_\theta(y_t, t), \tag{7}$$

where $Rpj_v$ is the Reprojection from $v$ perspective, and $f_\theta$ represents the denoiser network.

**Z-embedding Condition.** Even though the denoising formulation of diffusion modeling is to denoise the 2D noisy poses, the essential task of the denoiser focuses on removing noise from the 3D noisy poses under $t$ perturbed level. We leverage the geometric and experiment analyses to explore the relationship between 3D and 2D uncertainty (details in Appendix A). Our findings reveal that the 3D uncertainty range is not only influenced by the 2D uncertainty range but also relative to the depth z of the 3D point and the baseline width of the binocular setting. However, the diffused data in the fixed $t$ shares the same perturbed level, as defined in Eq. 2. To facilitate our denoiser to learn the noise within different uncertainty but under the same timestamp $t$, we introduce the $z_t$, the absolute depth of root joint of 3D poses $y_t$, as an additional condition. This is based on the assumption that the $z_t$ is close to the ground truth z because of the initial results with limited diffused. The denoiser is modified as:

$$y_\theta = f_\theta(y_t, z_t, t). \tag{8}$$

**Baseline-width-related Pose Normalization.** The depth z affects the noise along the x-axis, y-axis, and z-axis, while the baseline width only affects the noise on the z-axis. Considering the baseline width of binocular cameras will be changed in practical application. We propose a simple method to normalize and denormalize the 3D poses under different baseline width settings:

$$\bar{z}_t = Bz_t, \quad z_\theta = \bar{z}_\theta/B, \tag{9}$$

where $z_\theta$ shares the same definition with $z_t$, $B$ is a scalar representing the baseline width, and $\bar{z}$ is the normalized z. Through normalization and denormalization, the input to the denoiser is baseline width independent, but the 3D pose estimation results are not affected.

**Inference.** During the inference process, shown in Fig. 2(b), the initial binocular 2D poses $u_T$ estimated by a 2D detector are the input to our Dual-Diffusion model. Subsequently, the initial 3D pose $y_T$ is reconstructed and normalized, then fed into the denoiser to generate the plausible and accurate 3D pose $\tilde{y}_0$, along with their corresponding binocular 2D poses $\tilde{u}_0$. These poses $\tilde{u}_0$ are then diffused to $u_t$ using the DDIM strategy [37] for input to the next denoinsing. After iterative denoising for $K$ times. The final 3D pose $\tilde{y}_0$ and 2D poses $\tilde{u}_0$ are estimated. It should be noted that the 3D pose is converted to a root-relative format before being processed by the denoiser and then converted back afterward. For the experiments described below, we set $T = 25$ and $K = 1$ according to the ablation study in Sec. 4.3 and Appendix D.3. The denoiser follows a GCN-Transformer structure. Detailed architecture and training information can be found in Appendix B.

## 4 Experiments

Dual-Diffusion aims to reconstruct 3D pose from binocular 2D poses estimated from one off-the-shelf 2D detector. We refer to it as "Dual-Duffison-2D POSE DETECTOR" in tables. The baseline method for comparison is the Triangulation [9] (Tri), referred to as "Tri-2D POSE DETECTOR". The experiments are conducted on two benchmarks: 1) the short-baseline binocular benchmark, MHAD Berkeley dataset [27], and 2) the wide-baseline benchmark, H36M dataset [13]. MHAD [27] is a multi-modal dataset that encompasses 11 actions performed by 12 subjects. We choose the binocular camera pairs $(1 - 3, 2 - 4)$ in the L1 quad camera, with approximate $200mm$ baseline width. H36M [13] is one of the most popular datasets for 3D HPE. To simulate the binocular setup, the camera pairs $(1 - 3, 2 - 4)$ are selected, with about $3000mm$ baseline width. Mean Per Joint Position Error (MPJPE) is used to assess the accuracy, while Bone Length error (BL) and Symmetry error (Sym) are employed to evaluate the plausibility of 3D poses. Joint Detection Rate (JDR) is applied to assess 2D poses. The details of the metric and implementation can be found in Appendix C.

### 4.1 Comparison on MHAD

There are few methods designed particularly for binocular 3D HPE, except RSB-Pose [45]. We reproduce the state-of-the-art multiview 3D HPE methods and fine-tune them in binocular datasets, including TPPT [24], Epipolar_Tri [10], Algebraic_Tri [14] and AdaFuse [56]. Among them, RSB-Pose [45] utilizes Triangulation to lift 3D pose and refine it with Pose Transformer, Algebraic_Tri [14] employs a weighted Triangulation to reconstruct 3D pose. Other methods focus on estimating the 2D poses from fused features in a geometric or an attention mechanism based on the backbone ResNet [51], but the 3D pose is lifted solely through Triangulation. We use "METHOD NAME*" in tables to indicate the methods that primarily focus on improving multiview 2D pose estimation.

Table 1: **Quantitative Comparison on MHAD.** Scale is the resolution of image input to the 2D pose detector. The best results are highlighted in **bold**, and the second results are underlined. The results of the baseline comparison are in light blue , while the results of Dual-Diffusion are in dark blue .

| Method | Venue | 2D Pose Detector | Scale | MPJPE↓ (mm) | BL↓ (mm) | Sym↓ (mm) | JDR↑ (%) |
|---|---|---|---|---|---|---|---|
| TPPT [24] | ECCV'22 | TPPT* | 256 | 209.03 | 134.05 | 248.93 | - |
| RSB-Pose50 [45] | arXiv'23 | RSB-Pose50* | 256 | 32.10 | 10.21 | 12.13 | 96.62 |
| Epipolar_Tri [10] | CVPR'20 | Epipolar_Tri* | 256 | 90.73 | 33.67 | 34.21 | - |
| Dual-Diffusion-Epi | | Epipolar_Tri* | 256 | 76.42 14.31↓ | 27.02 6.65↓ | 26.42 7.79↓ | - |
| Tri-ViTPose | NeurIPS'22 | ViTPose [52] | 256 | 70.84 | 42.55 | 48.43 | 95.83 |
| Dual-Diffusion-ViT | | ViTPose [52] | 256 | 61.02 9.82↓ | 37.90 4.65↓ | 30.09 18.34↓ | 95.88 0.05↑ |
| Tri-ResNet50 | | ResNet50 [51] | 256 | 60.04 | 23.68 | 36.65 | 95.95 |
| Dual-Diffusion-ResNet50 | | ResNet50 [51] | 256 | 54.51 5.53↓ | 18.09 5.59↓ | 24.64 12.01↓ | 98.86 2.91↑ |
| Tri-RSB50 | | RSB-Pose50* | 256 | 35.40 | 11.36 | 14.25 | 96.62 |
| Dual-Diffusion-RSB50 | | RSB-Pose50* | 256 | 30.96 4.44↓ | 9.60 1.76↓ | 11.53 2.72↓ | 98.94 2.32↑ |
| Algebraic-Tri [14] | ICCV'19 | ResNet152 [51] | 384 | 51.69 | 27.11 | 45.69 | 95.95 |
| RSB-Pose152 [45] | arXiv'23 | RSB-Pose152* | 384 | 29.33 | 8.70 | 9.94 | 97.40 |
| AdaFuse [56] | IJCV'20 | AdaFuse* | 384 | 70.27 | 36.07 | 30.08 | 83.46 |
| Dual-Diffusion-Ada | | AdaFuse* | 256 | 53.77 16.50↓ | 24.59 11.48↓ | 23.19 6.89↓ | 95.37 11.91↑ |
| Tri-ResNet152 | | ResNet152 [51] | 384 | 48.26 | 19.22 | 27.73 | 95.95 |
| Dual-Diffusion-ResNet152 | | ResNet152 [51] | 384 | 43.57 4.69↓ | 16.20 3.02↓ | 14.91 12.82↓ | 97.26 1.31↑ |
| Tri-RSB152 | | RSB-Pose152* | 384 | 29.78 | 9.84 | 11.61 | 97.40 |
| Dual-Diffusion-RSB152 | | RSB-Pose152* | 384 | 27.76 2.02↓ | 7.56 2.28↓ | 9.83 1.78↓ | 99.20 1.80↑ |

To evaluate the performance of our Dual-Diffusion in 2D-3D lifting, we employ state-of-the-art binocular or multiview models as 2D detectors. Results are shown in Table 1. Our method generates more accurate and plausible 3D poses compared to the baseline. For instance, based on the 2D poses generated by ResNet50, Dual-Diffusion reduces the MPJPE by $5.53mm$ (a $9.2\%$ error reduction), and the BL and Sym by $5.59mm$ ($23.6\%$) and $12.01mm$ ($32.8\%$) respectively. This improvement is consistently observed with the 2D poses generated by RSB-Pose50* as well. When the image resolution increases to $384$, the 2D poses estimated are more accurate. Our method still outperforms the baseline with $2.02mm$ ($6.8\%$) in MPJPE, $2.28mm$ ($23.2\%$) in BL and $1.78mm$ ($15.3\%$) in Sym using the RSB-Pose152* 2D detector. Additionally, Dual-Diffusion generates more accurate 2D results, with increases of $2.32\%$ and $1.80\%$ compared to the initial results of 2D detectors RSB-Pose50* and RSB-Pose152*, respectively. The performance enhancement demonstrates our Dual-Diffusion can simultaneously denoise both 2D and 3D noisy poses and generate more accurate and plausible results. Regardless of the input resolution, our method consistently achieves the best results across all four metrics. This demonstrates the effectiveness of our approach in short-baseline binocular 3D HPE, making it promising for practical applications.

## 4.2 Comparison on H36M

In Table 2, the comparison is conducted on H36M [13]. Even though the uncertainty of initial 3D poses in the wide-baseline binocular is reduced compared to the short-baseline setup [45], our Dual-Diffusion still outperforms the baseline and achieves the best results across all metrics. Based on the RSB-Pose50* 2D detector, Dual-Diffusion

Table 2: **Quantitative Comparison on H36M.** Params is the number of model parameters excluding the backbone.

| Method | Scale | Params (M) | MPJPE↓ (mm) | BL↓ (mm) | Sym↓ (mm) | JDR↑ (%) |
|---|---|---|---|---|---|---|
| TPPT [24] | 256 | 9.70 | 40.72 | 22.49 | 25.44 | - |
| RSB-Pose50 [45] | 256 | 9.25 | 35.01 | 14.16 | 13.54 | 94.82 |
| Epipolar_Tri [10] | 256 | 0.08 | 41.22 | 20.39 | 20.18 | - |
| Dual-Diffusion-Epi | 256 | 0.74 | 37.03 4.19↓ | 16.90 3.49↓ | 18.08 2.10↓ | - |
| Tri-ViTPose | 256 | - | 41.49 | 18.09 | 20.75 | 93.33 |
| Dual-Diffusion-ViT | 256 | 0.74 | 35.20 6.29↓ | 16.02 2.07↓ | 19.66 1.09↓ | 95.77 2.44↑ |
| Tri-RSB50 | 256 | - | 38.13 | 17.26 | 16.82 | 94.82 |
| Dual-Diff-RSB50 | 256 | 0.74 | 33.17 4.96↓ | 12.29 4.97↓ | 11.75 5.07↓ | 94.91 0.09↑ |
| Algebraic-Tri [14] | 384 | 10.88 | 31.24 | 13.52 | 13.59 | 95.81 |
| RSB-Pose152 [45] | 384 | 9.25 | 30.07 | 13.33 | 12.86 | 95.93 |
| AdaFuse [56] | 384 | 1.02 | 30.27 | 15.23 | 14.36 | 94.25 |
| Dual-Diffusion-Ada | 384 | 0.74 | 29.17 1.10↓ | 13.85 1.38↓ | 13.57 0.79↓ | 96.06 1.81↑ |
| Tri-RSB152 | 384 | - | 30.54 | 13.65 | 13.42 | 95.93 |
| Dual-Diff-RSB152 | 384 | 0.74 | 28.67 1.87↓ | 12.06 1.59↓ | 12.35 1.07↓ | 95.97 0.04↑ |

achieves an error reduction of $13.0\%$, $28.8\%$, $30.1\%$ and $0.09\%$ in MPJPE, BL, Sym of 3D poses and JDR of 2D poses, respectively. Even at $384$ resolution, where other methods demonstrate effectiveness with about $30mm$ MPJPE results, our model still outperforms them by at least $1.87mm$ in MPJPE, achieving a result of $28.67mm$. It is demonstrated that Dual-Diffusion can enhance 3D poses under wide-baseline binocular configurations, showcasing its generalization capabilities.

Table 3: **Impact of Each Module.** Experiments are conducted on MHAD with 2D poses estimated from RSB-Pose152*. The first row is the result generated by Tri.

| Dual-Diff | Z-embedding | BaseL-norm | Params ↓ (M) | MACs ↓ (G) | MPJPE ↓ (mm) | BL ↓ (mm) | Sym ↓ (mm) |
|---|---|---|---|---|---|---|---|
| ✗ | ✗ | ✗ | - | - | 29.78 | 9.84 | 11.61 |
| ✓ | ✗ | ✗ | 0.74 | 0.42 | 28.20 1.58↓ | 8.81 1.03↓ | 11.23 0.38↓ |
| ✓ | ✓ | ✗ | 0.74 | 0.42 | 27.91 1.78↓ | 7.85 1.99↓ | 10.12 1.49↓ |
| ✓ | ✓ | ✓ | 0.74 | 0.42 | 27.76 2.02↓ | 7.56 2.28↓ | 9.83 1.78↓ |

Table 4: **Impact of BaseL-norm.** The results are MPJPE of 3D poses generated with 2D poses estimated from ResNet50.

| Baseline width (mm) | Tri | Dual-Diff | |
|---|---|---|---|
| | | ✗ BaseL-norm | ✓ BaseL-norm |
| 100 | 92.57 | 103.87 | 88.32 |
| 300 | 54.36 | 62.13 | 51.26 |

Table 5: **Comparison of Diffusion Models in MPJPE.** The 2D poses are estimated from RSB-Pose152*. $T$ is the overall diffusion steps.

| T (K=1) | 25 | 50 | 75 | 100 | 125 |
|---|---|---|---|---|---|
| random-noise | 328.82 | 270.51 | 234.41 | 135.72 | 70.19 |
| 2D-Diff | 29.40 | 29.55 | 28.81 | 31.09 | 29.25 |
| Dual-Diff | 28.20 | 28.31 | 28.17 | 28.19 | 28.25 |

Additionally, Dual-Diffusion requires only 0.74 million parameters, significantly fewer than other methods. This demonstrates that the effectiveness of pose refinement in our method does not rely on a large number of parameters, but rather on the dual diffusion modeling and training.

## 4.3 Ablation Study

The ablation experiments are conducted on MHAD, as the high uncertainty issue is more pronounced in short-baseline setups compared to wide-baseline ones [45].

**Impact of Each Module.** We first investigate the improvements provided by the pure Dual-Diffusion model described in Sec.3.2, denoted as "Dual-Diff" in the tables. Then, two additional modules, Z-embedding and BaseL-norm, are assessed. As depicted in Table3, the pure Dual-Diff significantly enhances the accuracy of 3D poses, achieving a 5.3% reduction in MPJPE and a 10.5% reduction in BL, indicative of its capability to generate more precise and plausible 3D poses. With the incorporation of Z-embedding as an additional condition, there are further 10.9% and 9.9% relative improvements in BL and Sym. It is worth noting that subjects often perform actions while tilting towards the camera rather than facing it, leading to differences in the depth of joints on the left and right sides, resulting in variations in their 3D uncertainty regions. Z-embedding is specifically designed to enhance the adaptability of the denoiser to different perturbed noise levels at the same time $t$, as demonstrated by the significant enhancement in Sym. Finally, the addition of the BaseL-norm brings slight improvements in three metrics. Additionally, the computational cost is limited.

**Zero-Shot 2D-3D Lifting.** The purpose of BaseL-norm is to enhance the flexibility of the Dual-Diffusion to various baseline width settings. To evaluate this, we adapt two additional camera pairs $(1-2, 1-4)$ on MHAD, representing binocular baseline width of $100mm$ and $300mm$, and conduct zero-shot 2D-3D lifting on them. Firstly, we employ ResNet50 to detect 2D poses. Then, the denoiser trained on $200mm$-baseline training dataset is directly utilized to generate 3D poses on $100mm$-baseline and $300mm$-baseline testing sets without fine-tuning. The results are presented in Table 4. Without BaseL-norm, the accuracy even worsens. But with BaseL-norm, there are $4.25mm$ and $3.1mm$ reductions in MPJPE for two settings, respectively. The improvement illustrates the zero-shot adaptability of the BaseL-norm module to various baseline widths, which is beneficial for further application in practice.

**Efficiency of the Uncertainty Distribution Initialization.** We use the initialized uncertainty distribution as the target distribution for diffusion instead of random noise. To evaluate its efficiency, we establish a baseline diffusion model that generates 3D poses by denoising random noise conditioned on 2D binocular poses, referred to as "random-noise". To mitigate the effect of camera projection matrices $P_v$, we translate 2D keypoints to 3D-aware vectors by $P_v^{-1}$. The comparison results are shown in Table 5. Under the same inference iteration $K = 1$, across the diffusion steps from 25 to 125, the random-noise consistently yields inferior results, while Dual-Diff achieves promising results even with $T = 25$. A smaller $T$ indicates a reduced time cost of training and inference.

Table 6: **Validation of 3D Uncertainty Distribution Modeling.** The results are MPJPE of 3D poses denoised by the denoiser trained with the MHAD training set.

| 2D Pose Detector | Dataset | 3D Pose | MPJPE (mm) | 2D Pose Detector | Dataset | 3D Pose | MPJPE (mm) |
|---|---|---|---|---|---|---|---|
| ResNet152 [51] | training | estimated | 15.93 | RSB-Pose152* [45] | training | estimated | 10.96 |
| | | GT+noise | 17.07 | | | GT+noise | 11.95 |
| | testing | estimated | 43.57 | | testing | estimated | 27.76 |
| | | GT+noise | 17.51 | | | GT+noise | 12.46 |

Table 7: **Comparison of Uncertainty Reconstructing and Uncertainty Statistics.** The 2D poses are estimated form ResNet152.

| Setting | Method | MPJPE (mm) | Setting | Method | MPJPE (mm) |
|---|---|---|---|---|---|
| training in small-dataset | Tri | 40.39 | training in large-dataset | Tri | 38.17 |
| and | Dual-Diff | 39.11 | and | Dual-Diff | 35.23 |
| testing in large-dataset | DiffPose [8] | 54.12 | testing in small-dataset | DiffPose [8] | 40.62 |

Moreover, there remains some ambiguity regarding whether the initial uncertainty distribution of the 3D estimation is effectively modeled. To investigate this, we perform denoising on simulated noisy 3D poses. Specifically, we first calculate the error between the 3D estimation and the 3D GT along each axis in the MHAD training set, storing this as the noise set. Then, we add noise sampled randomly from this set to the 3D GT along each axis. Finally, we use the denoiser to refine both the "GT + noise" and "estimated" 3D poses, comparing the MPJPE results. The hypothesis is that if the 3D uncertainty is well-modeled, the performance in refining both estimated 3D poses and simulated 3D poses should be similar. As shown in Table 6, regardless of the 2d pose detector, the accuracy of denoised "GT+noise" in both training and testing sets is all close to the "estimated" in the training set, demonstrating that the 3D uncertainty distribution is well-modeled.

**Uncertainty Reconstructing v.s. Uncertainty Statistics.** We argue that reconstructing 3D pose uncertainty from 2D pose uncertainty is a more practical approach. As discussed in Appendix A, the depth uncertainty of a 3D point increases with greater depth. Consequently, the statistical approach in DiffPose [8] tends to constrain the model to a narrow depth range, while Dual-Diffusion leverages more reliable 2D results. To evaluate this, we divide the MHAD into two subsets: large-dataset and small-dataset, based on the average depth of 3D poses and compare Dual-Diffusion with DiffPose. The results are illustrated in Table 7. Compared to the Triangulation baseline, performance improves with Dual-Diffusion but decreases with DiffPose. DiffPose suffers from the change of 3D poses uncertainty distribution while our Dual-Diffusion remains stable. This stability is due to the fact that Dual-Diffusion models the diffusion target using 2D uncertainty, which is significantly more stable than 3D uncertainty. The comparison of the stability in uncertainty between 2D poses and 3D poses can be found in Appendix D.3.

**3D Pose Priors v.s. 2D Pose Priors.** We argue that 3D pose priors are more easily captured compared to 2D pose priors because 2D poses vary under different perspectives. To validate this, we establish a diffusion model directly for denoising 2D poses, named "2D-Diff", and then reconstruct 3D poses. As shown in Table 5, Dual-Diff consistently outperforms 2D-Diff in terms of 3D pose accuracy. Furthermore, we compare the plausibility of 3D poses. As illustrated in Fig. 3, Dual-Diff exceeds 2D-Diff in both BL and Sym. These results collectively demonstrate the necessity of denoising in the 3D domain.

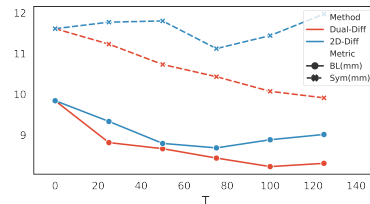

Figure 3: Dual-Diff (red) v.s. 2D-Diff (blue) under various $T$ and $K = 25$.

### 4.4 Visulization

**Dual-Diffusion Denoises the Binocular 2D Poses.** To understand the denoising process of binocular 2D poses, we conduct a simulation experiment. First, we gradually add noise to the ground truth 2D poses over $T = 25$ steps and generate the final noisy 2D poses. Then, we set the reverse iteration $K = 25$ to recurrently denoise the noisy poses. Fig. 4 illustrates the absolute distance between noisy data and the ground truth at each step $t$ in diffusion and each iteration $k$ in denoising. The noise added to the joint during the diffusion process is incrementally removed during the denoising process.

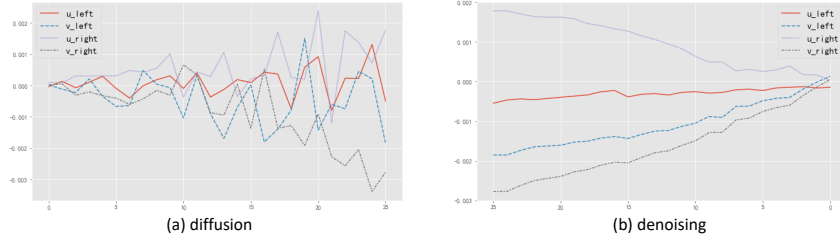

Figure 4: Step-wise errors of binocular 2D joints, $(u, v)_{left}$ and $(u, v)_{right}$, during the diffusion and denoising processes. The joint analyzed is the right knee.

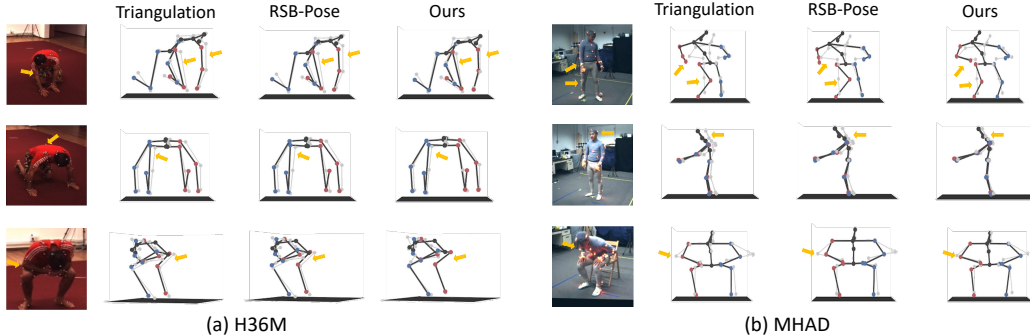

Figure 5: **Qualitative Comparison with Triangulation and RSB-Pose in 3D Pose Estimation.** 2D poses are estimated by RSB-Pose152*. The gray skeleton is the ground truth, while the black represents the estimates. Red and blue points correspond to joints on the right and left sides, respectively. Yellow arrows indicate parts of significant improvement achieved by our method.

**Dual-Diffusion Denoises the 3D Poses.** We provide a qualitative comparison with Tri-RSB152 and RSB-Pose152 in Fig. 5. Our Dual-Diffusion achieves more accurate 3D poses, particularly in cases of self-occlusion. For instance, in the $3^{rd}$ row of the $1^{st}$ column, the right hip is occluded, resulting in an inaccurate 3D pose using baseline Triangulation. Dual-Diffusion effectively corrects the 3D pose. This highlights our method's ability to denoise noisy 3D poses. Additionally, when baseline results are poor, such as in the $1^{st}$ row of the $2^{nd}$ column, RSB-Pose can only partially correct some joints, whereas our method corrects the entire right skeleton. More visualization is in Appendix D.4.

## 5 Conclusion and Discussion

This work introduces a novel framework, Dual-Diffusion, to reconstruct 3D poses from 2D poses estimated by off-the-shelf 2D pose detectors in a binocular configuration. Dual-Diffusion simultaneously denoises initial 2D and 3D poses within a single diffusion model. The diffusion process operates on 2D poses, while the denoising process occurs in 3D space, utilizing geometric mapping to connect the 2D and 3D domains. Comparisons with state-of-the-art methods demonstrate that our approach effectively denoises both 2D and 3D poses, yielding superior results and making it particularly suitable for binocular 3D HPE, especially in short-baseline configurations.

**Discussion.** We further extend the applicability of Dual-Diffusion to multiview settings, as shown in the Appendix D.2. The observed performance improvements validate the scalability of our method. Additionally, we incorporate 3D supervision (see Appendix D.3), revealing further advantages that will be explored in future work. However, we also acknowledge two main limitations of our method. Firstly, while simulation experiments are conducted to validate our hypothesis regarding 3D uncertainty factors (see Appendix A.2 for details), computing the boundary points of the 3D uncertainty region to explore it may lack rigor. We plan to leverage an algebraic approach to derive the range of the 3D uncertainty caused by 2D errors in future work. Secondly, our denoising network takes the root-relative 3D pose as input, neglecting optimization of the root joint. Although the root joint is finally refined (see Appendix D.1), another module to optimize the root joint is preferred.

## Acknowledgement

This work has been funded in part by the NSFC grants 62176156 and the Medical Engineering Cross Research Fund of Shanghai Jiao Tong University (YG2023ZD12).

## Footnotes

*Corresponding author: Xu Zhao.

## References

[1] Ijaz Akhter and Michael J Black. Pose-conditioned joint angle limits for 3d human pose reconstruction. In *Proceedings of the IEEE conference on computer vision and pattern recognition*, pages 1446–1455, 2015.

[2] Soubarna Banik, Alejandro Mendoza GarcÍa, and Alois Knoll. 3d human pose regression using graph convolutional network. In *2021 IEEE International Conference on Image Processing (ICIP)*, pages 924–928. IEEE, 2021.

[3] Magnus Burenius, Josephine Sullivan, and Stefan Carlsson. 3d pictorial structures for multiple view articulated pose estimation. In *2013 IEEE Conference on Computer Vision and Pattern Recognition*, pages 3618–3625. ISSN: 1063-6919.

[4] Xipeng Chen, Kwan-Yee Lin, Wentao Liu, Chen Qian, and Liang Lin. Weakly-supervised discovery of geometry-aware representation for 3d human pose estimation. In *Proceedings of the IEEE/CVF conference on computer vision and pattern recognition*, pages 10895–10904, 2019.

[5] Zhuo Chen, Xu Zhao, and Xiaoyue Wan. Structural triangulation: A closed-form solution to constrained 3d human pose estimation. In Shai Avidan, Gabriel Brostow, Moustapha Cissé, Giovanni Maria Farinella, and Tal Hassner, editors, *Computer Vision – ECCV 2022*, Lecture Notes in Computer Science, pages 695–711. Springer Nature Switzerland.

[6] Hai Ci, Mingdong Wu, Wentao Zhu, Xiaoxuan Ma, Hao Dong, Fangwei Zhong, and Yizhou Wang. Gfpose: Learning 3d human pose prior with gradient fields. In *Proceedings of the IEEE/CVF conference on computer vision and pattern recognition*, pages 4800–4810, 2023.

[7] Mohsen Gholami, Bastian Wandt, Helge Rhodin, Rabab Ward, and Z Jane Wang. Adaptpose: Cross-dataset adaptation for 3d human pose estimation by learnable motion generation. In *Proceedings of the IEEE/CVF Conference on Computer Vision and Pattern Recognition*, pages 13075–13085, 2022.

[8] Jia Gong, Lin Geng Foo, Zhipeng Fan, Qiuhong Ke, Hossein Rahmani, and Jun Liu. DiffPose: Toward more reliable 3d pose estimation. In *2023 IEEE/CVF Conference on Computer Vision and Pattern Recognition (CVPR)*, pages 13041–13051. IEEE.

[9] Richard Hartley and Andrew Zisserman. *Multiple View Geometry in Computer Vision*. Cambridge University Press, 2 edition, 2004.

[10] Yihui He, Rui Yan, Katerina Fragkiadaki, and Shoou-I Yu. Epipolar transformers. In *Proceedings of the ieee/cvf conference on computer vision and pattern recognition*, pages 7779–7788, 2020.

[11] Jonathan Ho, Ajay Jain, and Pieter Abbeel. Denoising diffusion probabilistic models. *Advances in neural information processing systems*, 33:6840–6851, 2020.

[12] Karl Holmquist and Bastian Wandt. Diffpose: Multi-hypothesis human pose estimation using diffusion models. In *Proceedings of the IEEE/CVF International Conference on Computer Vision*, pages 15977–15987, 2023.

[13] Catalin Ionescu, Dragos Papava, Vlad Olaru, and Cristian Sminchisescu. Human3. 6m: Large scale datasets and predictive methods for 3d human sensing in natural environments. *IEEE transactions on pattern analysis and machine intelligence*, 36(7):1325–1339, 2013.

[14] Karim Iskakov, Egor Burkov, Victor Lempitsky, and Yury Malkov. Learnable triangulation of human pose. In *Proceedings of the IEEE/CVF international conference on computer vision*, pages 7718–7727, 2019.

[15] Boyuan Jiang, Lei Hu, and Shihong Xia. Probabilistic triangulation for uncalibrated multi-view 3d human pose estimation. In *Proceedings of the IEEE/CVF International Conference on Computer Vision*, pages 14850–14860, 2023.

[16] Zhongyu Jiang, Zhuoran Zhou, Lei Li, Wenhao Chai, Cheng-Yen Yang, and Jenq-Neng Hwang. Back to optimization: Diffusion-based zero-shot 3d human pose estimation. In *Proceedings of the IEEE/CVF Winter Conference on Applications of Computer Vision*, pages 6142–6152, 2024.

[17] Diederik P Kingma and Max Welling. Auto-encoding variational bayes. *arXiv preprint arXiv:1312.6114*, 2013.

[18] Muhammed Kocabas, Salih Karagoz, and Emre Akbas. Self-supervised learning of 3d human pose using multi-view geometry. In *Proceedings of the IEEE/CVF conference on computer vision and pattern recognition*, pages 1077–1086, 2019.

[19] Nikos Kolotouros, Georgios Pavlakos, Dinesh Jayaraman, and Kostas Daniilidis. Probabilistic modeling for human mesh recovery. In *Proceedings of the IEEE/CVF international conference on computer vision*, pages 11605–11614, 2021.

[20] Jiefeng Li, Chao Xu, Zhicun Chen, Siyuan Bian, Lixin Yang, and Cewu Lu. HybrIK: A hybrid analytical-neural inverse kinematics solution for 3d human pose and shape estimation. In *2021 IEEE/CVF Conference on Computer Vision and Pattern Recognition (CVPR)*, pages 3382–3392. IEEE.

[21] Jiefeng Li, Chao Xu, Zhicun Chen, Siyuan Bian, Lixin Yang, and Cewu Lu. Hybrik: A hybrid analytical-neural inverse kinematics solution for 3d human pose and shape estimation. In *Proceedings of the IEEE/CVF conference on computer vision and pattern recognition*, pages 3383–3393, 2021.

[22] W Li, H Liu, H Tang, P Wang, and L MHFormer Van Gool. Multi-hypothesis transformer for 3d human pose estimation. arxiv 2021. *arXiv preprint arXiv:2111.12707*.

[23] Or Litany, Alex Bronstein, Michael Bronstein, and Ameesh Makadia. Deformable shape completion with graph convolutional autoencoders. In *Proceedings of the IEEE conference on computer vision and pattern recognition*, pages 1886–1895, 2018.

[24] Haoyu Ma, Zhe Wang, Yifei Chen, Deying Kong, Liangjian Chen, Xingwei Liu, Xiangyi Yan, Hao Tang, and Xiaohui Xie. Ppt: token-pruned pose transformer for monocular and multi-view human pose estimation. In *European Conference on Computer Vision*, pages 424–442. Springer, 2022.

[25] Julieta Martinez, Rayat Hossain, Javier Romero, and James J Little. A simple yet effective baseline for 3d human pose estimation. In *Proceedings of the IEEE international conference on computer vision*, pages 2640–2649, 2017.

[26] Alejandro Newell, Kaiyu Yang, and Jia Deng. Stacked hourglass networks for human pose estimation. In *Computer Vision–ECCV 2016: 14th European Conference, Amsterdam, The Netherlands, October 11-14, 2016, Proceedings, Part VIII 14*, pages 483–499. Springer, 2016.

[27] Ferda Ofli, Rizwan Chaudhry, Gregorij Kurillo, René Vidal, and Ruzena Bajcsy. Berkeley mhad: A comprehensive multimodal human action database. In *2013 IEEE workshop on applications of computer vision (WACV)*, pages 53–60. IEEE, 2013.

[28] Adam Paszke, Sam Gross, Soumith Chintala, Gregory Chanan, Edward Yang, Zachary DeVito, Zeming Lin, Alban Desmaison, Luca Antiga, and Adam Lerer. Automatic differentiation in pytorch. 2017.

[29] Georgios Pavlakos, Vasileios Choutas, Nima Ghorbani, Timo Bolkart, Ahmed AA Osman, Dimitrios Tzionas, and Michael J Black. Expressive body capture: 3d hands, face, and body from a single image. In *Proceedings of the IEEE/CVF conference on computer vision and pattern recognition*, pages 10975–10985, 2019.

[30] Georgios Pavlakos, Xiaowei Zhou, Konstantinos G. Derpanis, and Kostas Daniilidis. Coarse-to-fine volumetric prediction for single-image 3d human pose. In *2017 IEEE Conference on Computer Vision and Pattern Recognition (CVPR)*, pages 1263–1272. IEEE.

[31] Georgios Pavlakos, Xiaowei Zhou, Konstantinos G Derpanis, and Kostas Daniilidis. Harvesting multiple views for marker-less 3d human pose annotations. In *Proceedings of the IEEE conference on computer vision and pattern recognition*, pages 6988–6997, 2017.

[32] Haibo Qiu, Chunyu Wang, Jingdong Wang, Naiyan Wang, and Wenjun Zeng. Cross view fusion for 3d human pose estimation. In *Proceedings of the IEEE/CVF international conference on computer vision*, pages 4342–4351, 2019.

[33] Edoardo Remelli, Shangchen Han, Sina Honari, Pascal Fua, and Robert Wang. Lightweight multi-view 3d pose estimation through camera-disentangled representation. In *Proceedings of the IEEE/CVF conference on computer vision and pattern recognition*, pages 6040–6049, 2020.

[34] Cédric Rommel, Eduardo Valle, Mickaël Chen, Souhaiel Khalfaoui, Renaud Marlet, Matthieu Cord, and Patrick Pérez. Diffhpe: Robust, coherent 3d human pose lifting with diffusion. In *Proceedings of the IEEE/CVF International Conference on Computer Vision*, pages 3220–3229, 2023.

[35] Wenkang Shan, Zhenhua Liu, Xinfeng Zhang, Shanshe Wang, Siwei Ma, and Wen Gao. P-stmo: Pre-trained spatial temporal many-to-one model for 3d human pose estimation. In *European Conference on Computer Vision*, pages 461–478. Springer, 2022.

[36] Wenkang Shan, Zhenhua Liu, Xinfeng Zhang, Zhao Wang, Kai Han, Shanshe Wang, Siwei Ma, and Wen Gao. Diffusion-based 3d human pose estimation with multi-hypothesis aggregation. In *Proceedings of the IEEE/CVF International Conference on Computer Vision*, pages 14761–14771, 2023.

[37] Jiaming Song, Chenlin Meng, and Stefano Ermon. Denoising diffusion implicit models. *arXiv preprint arXiv:2010.02502*, 2020.

[38] Yang Song and Stefano Ermon. Generative modeling by estimating gradients of the data distribution. *Advances in neural information processing systems*, 32, 2019.

[39] Yang Song, Jascha Sohl-Dickstein, Diederik P Kingma, Abhishek Kumar, Stefano Ermon, and Ben Poole. Score-based generative modeling through stochastic differential equations. *arXiv preprint arXiv:2011.13456*, 2020.

[40] Ke Sun, Bin Xiao, Dong Liu, and Jingdong Wang. Deep high-resolution representation learning for human pose estimation. In *Proceedings of the IEEE/CVF conference on computer vision and pattern recognition*, pages 5693–5703, 2019.

[41] Zhenhua Tang, Zhaofan Qiu, Yanbin Hao, Richang Hong, and Ting Yao. 3d human pose estimation with spatio-temporal criss-cross attention. In *Proceedings of the IEEE/CVF Conference on Computer Vision and Pattern Recognition*, pages 4790–4799, 2023.

[42] Bugra Tekin, Isinsu Katircioglu, Mathieu Salzmann, Vincent Lepetit, and Pascal Fua. Structured prediction of 3d human pose with deep neural networks. *arXiv preprint arXiv:1605.05180*, 2016.

[43] Denis Tome, Chris Russell, and Lourdes Agapito. Lifting from the deep: Convolutional 3d pose estimation from a single image. In *Proceedings of the IEEE conference on computer vision and pattern recognition*, pages 2500–2509, 2017.

[44] Ashish Vaswani, Noam Shazeer, Niki Parmar, Jakob Uszkoreit, Llion Jones, Aidan N Gomez, Łukasz Kaiser, and Illia Polosukhin. Attention is all you need. *Advances in neural information processing systems*, 30, 2017.

[45] Xiaoyue Wan, Zhuo Chen, Yiming Bao, and Xu Zhao. Rsb-pose: Robust short-baseline binocular 3d human pose estimation with occlusion handling. *arXiv preprint arXiv:2311.14242*, 2023.

[46] Xiaoyue Wan, Zhuo Chen, and Xu Zhao. View consistency aware holistic triangulation for 3d human pose estimation. *Computer Vision and Image Understanding*, 236:103830, 2023.

[47] Bastian Wandt and Bodo Rosenhahn. Repnet: Weakly supervised training of an adversarial reprojection network for 3d human pose estimation. In *Proceedings of the IEEE/CVF conference on computer vision and pattern recognition*, pages 7782–7791, 2019.

[48] Jian Wang, Lingjie Liu, Weipeng Xu, Kripasindhu Sarkar, and Christian Theobalt. Estimating egocentric 3d human pose in global space. In *Proceedings of the IEEE/CVF International Conference on Computer Vision*, pages 11500–11509, 2021.

[49] Jinbao Wang, Shujie Tan, Xiantong Zhen, Shuo Xu, Feng Zheng, Zhenyu He, and Ling Shao. Deep 3d human pose estimation: A review. *Computer Vision and Image Understanding*, 210:103225, 2021.

[50] Tom Wehrbein, Marco Rudolph, Bodo Rosenhahn, and Bastian Wandt. Probabilistic monocular 3d human pose estimation with normalizing flows. In *Proceedings of the IEEE/CVF international conference on computer vision*, pages 11199–11208, 2021.

[51] Bin Xiao, Haiping Wu, and Yichen Wei. Simple baselines for human pose estimation and tracking. In *Proceedings of the European conference on computer vision (ECCV)*, pages 466–481, 2018.

[52] Yufei Xu, Jing Zhang, Qiming Zhang, and Dacheng Tao. Vitpose: Simple vision transformer baselines for human pose estimation. *Advances in Neural Information Processing Systems*, 35:38571–38584, 2022.

[53] Youze Xue, Jiansheng Chen, Xiangming Gu, Huimin Ma, and Hongbing Ma. Boosting monocular 3d human pose estimation with part aware attention. *IEEE Transactions on Image Processing*, 31:4278–4291, 2022.

[54] Petrissa Zell, Bastian Wandt, and Bodo Rosenhahn. Joint 3d human motion capture and physical analysis from monocular videos. In *Proceedings of the IEEE Conference on Computer Vision and Pattern Recognition Workshops*, pages 17–26, 2017.

[55] Jinlu Zhang, Zhigang Tu, Jianyu Yang, Yujin Chen, and Junsong Yuan. Mixste: Seq2seq mixed spatio-temporal encoder for 3d human pose estimation in video. In *Proceedings of the IEEE/CVF conference on computer vision and pattern recognition*, pages 13232–13242, 2022.

[56] Zhe Zhang, Chunyu Wang, Weichao Qiu, Wenhu Qin, and Wenjun Zeng. Adafuse: Adaptive multiview fusion for accurate human pose estimation in the wild. *International Journal of Computer Vision*, 129:703–718, 2021.

[57] Long Zhao, Xi Peng, Yu Tian, Mubbasir Kapadia, and Dimitris N Metaxas. Semantic graph convolutional networks for 3d human pose regression. In *Proceedings of the IEEE/CVF conference on computer vision and pattern recognition*, pages 3425–3435, 2019.

[58] Qitao Zhao, Ce Zheng, Mengyuan Liu, and Chen Chen. A single 2d pose with context is worth hundreds for 3d human pose estimation. In *Thirty-seventh Conference on Neural Information Processing Systems*, 2023.

[59] Qitao Zhao, Ce Zheng, Mengyuan Liu, Pichao Wang, and Chen Chen. Poseformerv2: Exploring frequency domain for efficient and robust 3d human pose estimation. In *Proceedings of the IEEE/CVF Conference on Computer Vision and Pattern Recognition*, pages 8877–8886, 2023.

[60] Weixi Zhao, Yunjie Tian, Qixiang Ye, Jianbin Jiao, and Weiqiang Wang. Graformer: Graph convolution transformer for 3d pose estimation. *arXiv preprint arXiv:2109.08364*, 2021.

[61] Jieming Zhou, Tong Zhang, Zeeshan Hayder, Lars Petersson, and Mehrtash Harandi. Diff3dhpe: A diffusion model for 3d human pose estimation. In *Proceedings of the IEEE/CVF International Conference on Computer Vision*, pages 2092–2102, 2023.

[62] Xingyi Zhou, Qixing Huang, Xiao Sun, Xiangyang Xue, and Yichen Wei. Towards 3d human pose estimation in the wild: a weakly-supervised approach. In *Proceedings of the IEEE international conference on computer vision*, pages 398–407, 2017.

# A Relationship between 3D and 2D Uncertainty

## A.1 Theoretical Analysis

We simplify the relationship between 3D and 2D uncertainty to the relationship between 3D uncertainty range $\Delta x, \Delta z, \Delta z'$ with 2D uncertainty range $\Delta u$. We analyze it in the x-o-z cross-section of 3D space, as shown in Fig. 6. $O_l$ and $O_r$ are the optical centers of binocular cameras which are rectified and their image plane is parallel to the x-axis. The red point $H$ is the ground truth of a 3D point with depth z. The analysis is based on the hypothesis that the uncertainty range of the estimated 2D point is the same in 2 views with $\Delta u$ along the x-axis. The ground truth 2D point is the projection of $H$.

Point $A$ represents the intersection of the left boundaries of the 2D uncertainty regions from both viewpoints and serves as the left boundary along the x-axis in the 3D uncertainty region. $B$ denotes the intersection of the right boundaries from both viewpoints, forming the right boundary along the x-axis in the 3D space. $C$ is the intersection between the right boundary of the 2D uncertainty region from the left view and the left boundary of the 2D uncertainty region from the right view, establishing the lower bound along the z-axis of 3D uncertainty. Conversely, point $D$ marks the upper bound along the z-axis.

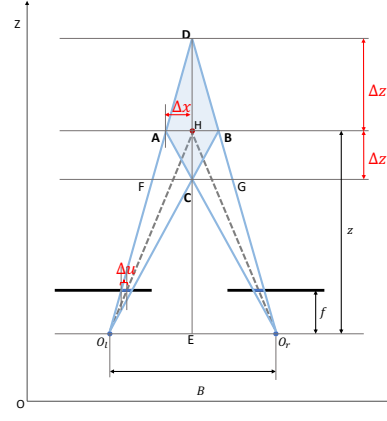

Figure 6: 3D reconstructing uncertainty range of binocular configuration.

**Uncertainty along the x-axis.** To analysis the relationship between $\Delta x$ and $\Delta u$, we first prove that line segment $AB$ is parallel to the image plain. Based on the disparity formula $z = fB/\Delta d$, $A$ and $H$ have the same depth z since they share the same disparity, and similarly for points $B$ and $H$. Therefore, line segment $AB$ passes through $H$ and is parallel to the image plane. Then, the segment $AH$ can be derived using similar triangles:

$$\frac{\Delta x}{\Delta u} = \frac{z}{f}. \tag{10}$$

**Uncertainty along the z-axis.** We first prove that the midpoint $E$ of $O_lO_r$ and H lie on the line $DC$. Through point $C$, draw a line parallel to the image plane, intersecting $O_lD$ and $O_rD$ at points $F$ and $G$ respectively. Line $DC$ intersects $AB$ and $O_rO_l$ at points $E'$ and $H'$. According to the properties of similar triangles, $FC = GC$. Furthermore, cause $\triangle DFC \sim \triangle DO_lE'$ and $\triangle DGC \sim \triangle DO_rE'$, we can deduce that $O_lE' = O_rE'$. Similarly, it can be proven that $AH' = BH'$. Then, the segment $\Delta z$ can be derived using similar triangles:

$$\frac{\Delta z}{\Delta z + z} = \frac{\Delta x}{B/2}. \tag{11}$$

Combining with Eq. 10, it can be written as:

$$\frac{1}{\Delta z} = \frac{Bf}{2z^2}\frac{1}{\Delta u} - \frac{1}{z} \tag{12}$$

The segment $\Delta z'$ can be derived in the similar way:

$$\frac{1}{\Delta z'} = \frac{Bf}{2z^2}\frac{1}{\Delta u} + \frac{1}{z} \tag{13}$$

We omit the proof of $\Delta y$ as it follows the same reasoning as $\Delta x$, given that the standard deviation along the u-axis and v-axis in a 2D image plane are equivalent. In summary, the factors contributing to the 3D uncertainty range include the 2D uncertainty range, the depth of the 3D point, and the baseline width. The 3D depth affects uncertainty along the x-axis, y-axis, and z-axis. The baseline width only affects the z-axis uncertainty.

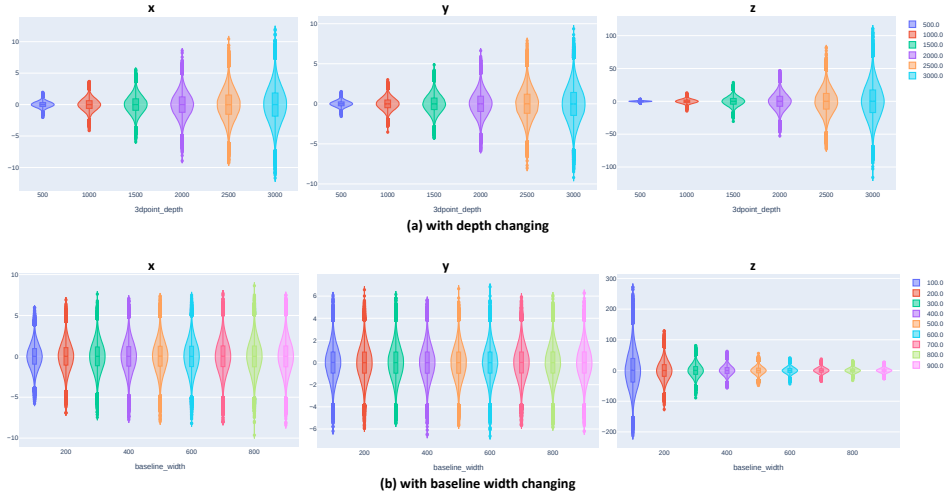

Figure 7: Uncertainty distribution of a 3D point as (a) depth or (b) baseline width changes.

## A.2 Experiment Analysis

The analysis above solely relies on the boundary points to explore the relationship between 3D and 2D uncertainty, which is not rigorous. Hence, we conduct the experiments to validate the conclusion.

We change the point depth from $500mm$ to $3000mm$ with $500mm$ step to discover the relationship between 3D uncertainty and depth. The binocular 2D points within a fixed uncertainty range are sampled randomly and then reconstructed as a 3D point using Triangulation. We sample for $100000$ times at each depth and then analyze the 3D uncertainty range, visualized in Fig. 7(a). The 3D depth affects the uncertainty of a 3D point along the x-axis, y-axis, and z-axis but with different relationships. Hence, we add z-embedding as an additional condition to guide the denoiser. The experiment investigating the impact of baseline width is depicted in Fig.7(b). Baseline width solely affects the uncertainty distribution along the z-axis. As depicted in Eq.12 and Eq.13, aside from the constant term, we introduce a normalization factor $B$ to standardize the error distribution across different baseline widths. The distribution with and without normalization is illustrated in Fig. 8.

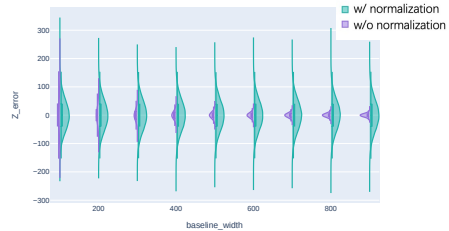

Figure 8: The difference in z-axis uncertainty distribution between with and without normalization as baseline width changes.

## B Denoiser Architecture and Training Details

**Denoiser Architecture.** The denoiser architecture, illustrated in Fig. 9, is similar to the one proposed by [8] and essentially follows a GCN-Transformer structure. The 3D pose, represented as $J$ joint locations, is first input into a GCN layer. The graph for the GCN is defined by the skeleton connections, which encode the topological features of each joint. Next, five stacked GCN-Attention modules are used to enhance the global-local perception of the joint features. Each GCN-Attention module comprises one Attention layer and two GCN layers. Each Attention layer includes a 4-head self-attention mechanism. Timestamp embedding and Z-embedding are added to each GCN-Attention module. The embedding schedule is sinusoidal [44]. Finally, the 3D pose is output after passing through another GCN layer.

**Training Details.** The training process is depicted in Fig. 9. We first generate diffused data for training as follows: 1) Initialization: Start with the ground truth 2D poses $u_0$. 2) Noise Addition:

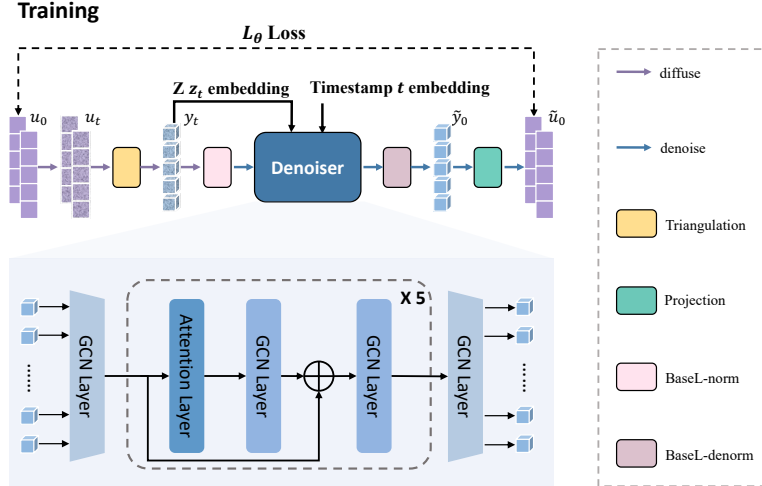

Figure 9: An overall training process and denoiser architecture for Dual-Diffusion.

Randomly choose a timestamp $t \in [1, T]$, add noise to generate noisy 2D poses $u_t$. 3) Triangulation: Use triangulation to reconstruct the noisy 3D poses $y_t$. Then we denoise to recover the data through the denoiser: 1) Denoising 3D Poses: The denoiser takes the noisy 3D poses $y_t$ as input and denoises them to recover original 3D poses $\tilde{y}_0$. BaseL-norm is applied before the denoiser and BaseL-denorm is applied after. 2) Reprojection: Reproject the denoised 3D poses back to 2D poses $\tilde{u}_0$. The training objective follows the $L_\theta$ in Eq. 7 to facilitate the denoiser recovering the original input.

The Dual-Diffusion is implemented by PyTorch [28] using Adam optimizer with learning rate $0.00002$ and other parameters are all default. We train our Dual-Diffusion in MHAD [27] for 80 epochs, and in H36M [13] for 40 epochs, respectively. There is no other schedule for the learning rate employed. All experiments are conducted on GeForce RTX 2080 Ti.

## C  More Details of Experiment Implementation

**Evaluation Metrics.**  To evaluate 3D pose accuracy, we utilize Mean Per Joint Position Error (MPJPE). Besides, we employ two metrics to assess the plausibility of the 3D pose: Bone Length error (BL), which measures the average distance between the predicted and ground truth bone lengths; and Symmetry (Sym) [34], which quantifies the average difference between the lengths of corresponding left and right bones. The Joint Detection Rate (JDR) is applied for 2D pose evaluation with the detection threshold set at $2.5\%$ of the bounding box width.

**Implementation Details.**  We choose ResNet [51] and RSB-Pose* [45] as our 2D pose detectors and divide them into two categories with different input image resolutions. To acquire the initial uncertainty distribution of estimated 2D poses, the methods are distinct for different 2D detector. For ResNet, the standard deviation are set to 2 and 3 separately for 256 and 384 image resolution, as the training objective formulating. For RSB-Pose*, we obtain the standard deviation by statistically analyzing the estimation results on the training set. Because its training objective is MPJPE but not heatmap. Then, the standard deviation is scaled by image resolution to acquire $\sigma_T$. The diffusion step $T$ is set to 25, and the reverse step $K$ is set to 1 decided by the ablation study in Sec. 4.3 and Appendix D.3. To acquire the deterministic results, the reverse sampling is followed to the ODEs [39].

## D  More Experiments

### D.1  Comparison on MHAD

We compare the per-joint error with the baseline using the same 2D pose detector. The results are shown in Table 8. Our Dual-Diffusion method surpasses the baseline in most joints, particularly

Table 8: **Quantitative Comparison of Per-Joint Error on MHAD.** Scale is the resolution of image input to the 2D pose detector. The column in green is the root joint.

| MPJPE (mm) | Scale | Shlder | Elbow | Wrist | Hip | Knee | Ankle | Pelvi | Belly | Neck | Nose | Head | Avg. |
|---|---|---|---|---|---|---|---|---|---|---|---|---|---|
| Tri-RSB50 | 256 | 29.52 | 41.76 | 49.20 | 25.12 | 22.56 | 24.95 | 103.20 | 25.19 | 29.20 | 27.98 | 30.68 | 35.40 |
| Dual-Diffusion-RSB50 | 256 | 26.18 | 39.70 | 51.80 | 24.21 | 21.54 | 30.54 | 40.33 | 23.12 | 25.11 | 21.46 | 26.40 | 30.96 |
| Tri-RSB152 | 384 | 29.06 | 37.81 | 48.56 | 25.02 | 17.85 | 23.46 | 41.54 | 24.30 | 25.66 | 23.35 | 28.83 | 29.78 |
| Dual-Diffusion-RSB152 | 384 | 26.26 | 35.95 | 47.56 | 23.72 | 17.09 | 21.15 | 31.22 | 23.07 | 23.95 | 22.79 | 28.09 | 27.76 |

Table 9: **Applicability to Multiview Settings.** The results are MPJPE of 3D poses denoised from the initial estimated 3D poses using ResNet152 in 2-view, 3-view, and 4-view H36M testing sets.

| Method | View Number | MPJPE (mm) | BL (mm) | Sym (mm) | JDR (%) | View Number | MPJPE (mm) | BL (mm) | Sym (mm) | JDR (%) | View Number | MPJPE (mm) | BL (mm) | Sym (mm) | JDR (%) |
|---|---|---|---|---|---|---|---|---|---|---|---|---|---|---|---|
| Tri-ResNet152 | 2 | 31.51 | 14.38 | 16.29 | 95.81 | 3 | 30.13 | 13.26 | 15.75 | 96.46 | 4 | 29.93 | 13.14 | 14.57 | 94.96 |
| Dual-DIff-ResNet152 | | 29.15 | 12.06 | 13.37 | 95.92 | | 28.69 | 11.99 | 12.42 | 96..60 | | 28.44 | 12.22 | 12.39 | 95.21 |

Table 10: **Impact of Baseline Width to Uncertainty.**

| Baseline width (mm) | 2D MPJPE (pixel) | 3D MPJPE (mm) |
|---|---|---|
| 100 | 3.084 | 120.37 |
| 200 | 3.079 | 63.18 |
| 300 | 3.079 | 54.69 |

Table 11: **Impact of Depth to Uncertainty.** The 2D poses are estimated by ResNet152. STD is the standard deviation.

| Dateset | 2D MPJPE (pixel) | 3D MPJPE (mm) | 3D MPJPE STD (mm) |
|---|---|---|---|
| large-dataset | 5.26 | 40.39 | 1025.88 |
| small-dataset | 5.33 | 38.17 | 161.88 |

the root joints and those close to the root. Interestingly, even though the input to the denoiser is the root-relative 3D pose, the root joint itself is still optimized as highlighted in green. We hypothesize that this optimization occurs due to the correlation established with the connected joints, which demonstrates that the pose prior is effectively captured.

## D.2 Applicability to Multiview Settings

To evaluate the applicability of Dual-Diffusion in multiview settings, we conduct experiments by denoising the initially estimated 3D poses using ResNet152 across 2-view, 3-view, and 4-view configurations of the H36M testing set. The results, as shown in Table 9, demonstrate that Dual-Diffusion improves performance compared to the Triangulation baseline. However, the degree of improvement decreases as the number of views increases. This aligns with our explanation that 3D uncertainty is more ambiguous in binocular settings than in 3-view or 4-view setups, where 3D uncertainty is reduced, making denoising less impactful.

## D.3 Ablation Study

**Uncertainty Stability Comparison.** We conducted a comparison of the stability in uncertainty between 2D and 3D poses. First, we evaluated the 2D MPJPE and 3D MPJPE estimated by ResNet50 under binocular settings with varying baseline widths, as shown in Table 10. Additionally, we compared the 2D and 3D uncertainties obtained from ResNet152 across different depths, detailed in Table 11. The results clearly demonstrate that 2D pose estimation is more robust than 3D pose estimation.

**Impact of the Inference Iteration Time.** To explore the impact of the inference iteration time $K$, we denoise the 3D poses for $K \in \{1, 2, 5, 10, 20\}$ iterations and compare the MPJPE results. When $K = 1$, we achieve the best result, as shown in Table 12. Additionally, our Dual-Diff outperforms the 2D-Diff across all $K$, demonstrating the effectiveness of modeling pose priors in the 3D domain.

**Impact of Supervision.** We implement experiments by using 3D supervision alone and in combination with 2D supervision to retrain

Table 12: **Impact of Inference Iteration Times K.** The results are based on RSB-Pose152*.

| K (T=25) | 1 | 2 | 5 | 10 | 20 |
|---|---|---|---|---|---|
| Dual-Diff | 28.20 | 28.36 | 28.80 | 28.97 | 29.08 |
| 2D-Diff | 29.40 | 33.75 | 35.75 | 39.15 | 40.30 |

Table 13: **Impact of Supervision.** The experiments are conducted in MHAD, based on RSB-Pose152*.

| Loss | MPJPE (mm) | BL (mm) | Sym (mm) |
|---|---|---|---|
| 2D | 27.76 | 7.56 | 9.83 |
| 3D | 29.38 | 8.62 | 10.57 |
| 2D+3D | 27.73 | 7.43 | 9.43 |

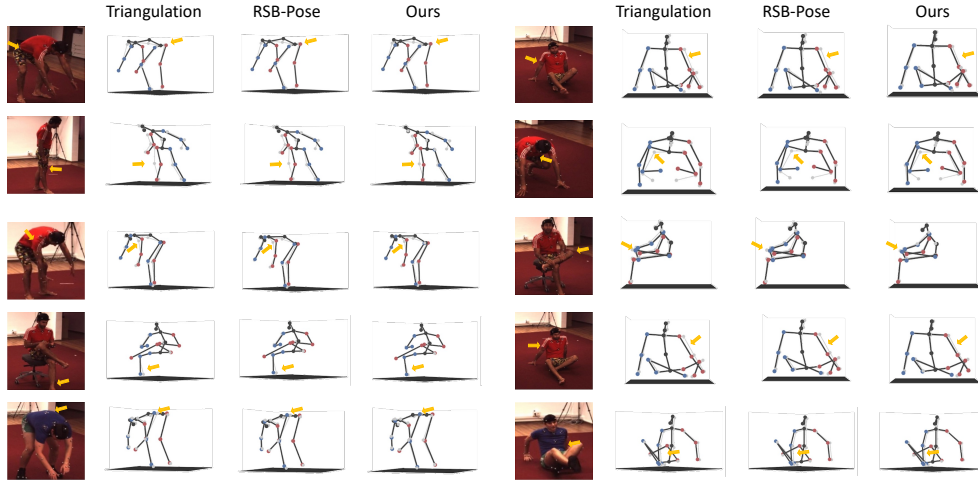

Figure 10: **Qualitative Comparison on H36M.** 2D poses are estimated by RSB-Pose152*. The gray skeleton is the ground truth, while the black represents the estimates. Red and blue points correspond to joints on the right and left sides, respectively. Yellow arrows indicate parts of significant improvement achieved by our method.

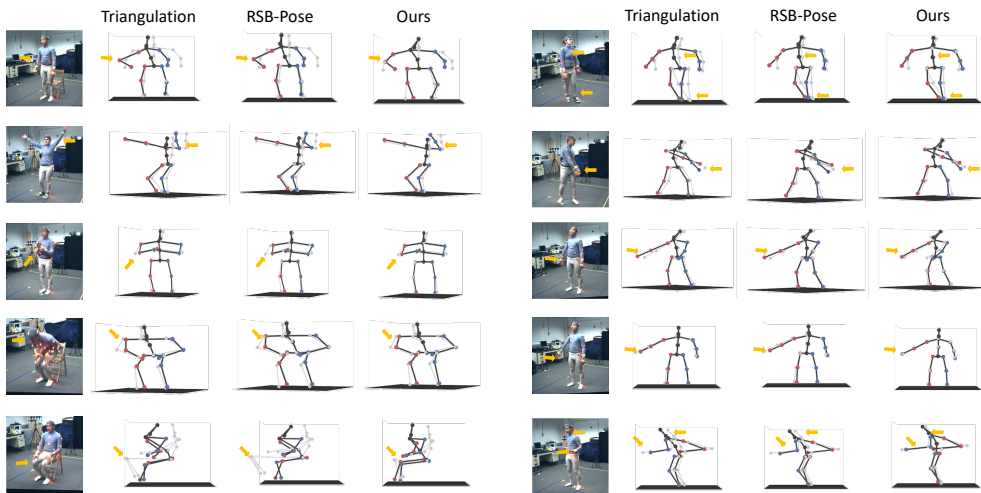

Figure 11: **Qualitative Comparison on MHAD.** 2D poses are estimated by RSB-Pose152*. The gray skeleton is the ground truth, while the black represents the estimates. Red and blue points correspond to joints on the right and left sides, respectively. Yellow arrows indicate parts of significant improvement achieved by our method.

Dual-Diffusion, based on the RSB152 backbone. As shown in Table 13, adding 3D supervision additionally enhances 3D accuracy and plausibility. Considering that Dual-Diffusion still has room for improvement, future work will focus on addressing the limitations discussed in the paper and exploring additional enhancement methods.

## D.4    Visualization

We provide a qualitative comparison with Triangulation and RSB-Pose on H36M and MHAD, as illustrated in Fig.10 and Fig.11 respectively. Our Dual-Diffusion method noticeably corrects self-occluded joints in MHAD. In the wide-baseline H36M, although the joints occluded in both views are minimal, our method still achieves enhancements compared to Triangulation and RSB-Pose.

